# Identifying Patients at Risk of Major Adverse Cardiovascular Events Using Symbolic Mismatch

**Zeeshan Syed**
University of Michigan
Ann Arbor, MI 48109
zhs@eecs.umich.edu

**John Guttag**
Massachusetts Institute of Technology
Cambridge, MA 02139
guttag@csail.mit.edu

## Abstract

Cardiovascular disease is the leading cause of death globally, resulting in 17 million deaths each year. Despite the availability of various treatment options, existing techniques based upon conventional medical knowledge often fail to identify patients who might have benefited from more aggressive therapy. In this paper, we describe and evaluate a novel unsupervised machine learning approach for cardiac risk stratification. The key idea of our approach is to avoid specialized medical knowledge, and assess patient risk using *symbolic mismatch*, a new metric to assess similarity in long-term time-series activity. We hypothesize that high risk patients can be identified using symbolic mismatch, as individuals in a population with unusual long-term physiological activity. We describe related approaches that build on these ideas to provide improved medical decision making for patients who have recently suffered coronary attacks. We first describe how to compute the symbolic mismatch between pairs of long term electrocardiographic (ECG) signals. This algorithm maps the original signals into a symbolic domain, and provides a quantitative assessment of the difference between these symbolic representations of the original signals. We then show how this measure can be used with each of a one-class SVM, a nearest neighbor classifier, and hierarchical clustering to improve risk stratification. We evaluated our methods on a population of 686 cardiac patients with available long-term electrocardiographic data. In a univariate analysis, all of the methods provided a statistically significant association with the occurrence of a major adverse cardiac event in the next 90 days. In a multivariate analysis that incorporated the most widely used clinical risk variables, the nearest neighbor and hierarchical clustering approaches were able to statistically significantly distinguish patients with a roughly two-fold risk of suffering a major adverse cardiac event in the next 90 days.

## 1 Introduction

In medicine, as in many other disciplines, decisions are often based upon a comparative analysis. Patients are given treatments that worked in the past on apparently similar conditions. When given simple data (e.g., demographics, comorbidities, and laboratory values) such comparisons are relatively straightforward. For more complex data, such as continuous long-term signals recorded during physiological monitoring, they are harder. Comparing such time-series is made challenging by three factors: the need to efficiently compare very long signals across a large number of patients, the need to deal with patient-specific differences, and the lack of *a priori* knowledge associating signals with long-term medical outcomes.

In this paper, we exploit three different ideas to address these problems.

- We address the problems related to scale by abstracting the raw signal into a sequence of symbols,

- We address the problems related to patient-specific differences by using a novel technique, *symbolic mismatch*, that allows us to compare sequences of symbols drawn from distinct alphabets. Symbolic mismatch compares long-term time-series by quantifying differences between the morphology and frequency of prototypical functional units, and

- We address the problems related to lack of *a priori* knowledge using three different methods, each of which exploits the observation that high risk patients typically constitute a small minority in a population.

In the remainder of this paper, we present our work in the context of risk stratification for cardiovascular disease. Cardiovascular disease is the leading cause of death globally and causes roughly 17 million deaths each year [3]. Despite improvements in survival rates, in the United States, one in four men and one in three women still die within a year of a recognized first heart attack [4]. This risk of death can be substantially lowered with an appropriate choice of treatment (e.g., drugs to lower cholesterol and blood pressure; operations such as coronary artery bypass graft; and medical devices such as implantable cardioverter defibrillators) [3]. However, matching patients with treatments that are appropriate for their risk has proven to be challenging [5,6].

That existing techniques based upon conventional medical knowledge have proven inadequate for risk stratification leads us to explore methods with few *a priori* assumptions. We focus, in particular, on identifying patients at elevated risk of major adverse cardiac events (death, myocardial infarction and severe recurrent ischemia) following coronary attacks. This work uses long-term ECG signals recorded during patient admission for ACS. These signals are routinely collected, potentially allowing for the work presented here to be deployed easily without imposing additional needs on patients, caregivers, or the healthcare infrastructure.

Fortunately, only a minority of cardiac patients experience serious subsequent adverse cardiovascular events. For example, cardiac mortality over a 90 day period following acute coronary syndrome (ACS) was reported to be 1.79% for the SYMPHONY trial involving 14,970 patients [1] and 1.71% for the DISPERSE2 trial with 990 patients [2]. The rate of myocardial infarction (MI) over the same period for the two trials was 5.11% for the SYMPHONY trial and 3.54% for the DISPERSE2 trial. Our hypothesis is that these patients can be discovered as anomalies in the population, i.e., their physiological activity over long periods of time is dissimilar to the majority of the patients in the population. In contrast to algorithms that require labeled training data, we propose identifying these patients using unsupervised approaches based on three machine learning methods previously reported in the literature: one-class support vector machines (SVMs), nearest neighbor analysis, and hierarchical clustering.

The main contributions of our work are: (1) we describe a novel unsupervised approach to cardiovascular risk stratification that is complementary to existing clinical approaches, (2) we explore the idea of similarity-based clinical risk stratification where patients are categorized in terms of their similarities rather than specific features based on prior knowledge, (3) we develop the hypothesis that patients at future risk of adverse outcomes can be detected using an unsupervised approach as outliers in a population, (4) we present symbolic mismatch, as a way to efficiently compare very long time-series without first reducing them to a set of features or requiring symbol registration across patients, and (5) we present a rigorous evaluation of unsupervised similarity-based risk stratification using long-term data from nearly 700 patients with detailed admissions and follow-up data.

## 2    Symbolic Mismatch

We start by describing the process through which symbolic mismatch is measured on ECG signals.

### 2.1    Symbolization

As a first step, the ECG signal $z_m$ for each patient $m = 1, ..., n$ is symbolized using the technique proposed by [7]. To segment the ECG signal into beats, we use two open-source QRS detection algorithms [8,9]. QRS complexes are marked at locations where both algorithms agree. A variant of dynamic time-warping (DTW) [7] is then used to quantify differences in morphology between

beats. Using this information, beats with distinct morphologies are partitioned into groups, with each group assigned a unique label or symbol. This is done using a Max-Min iterative clustering algorithm that starts by choosing the first observation as the first centroid, $c_1$, and initializes the set $S$ of centroids to $\{c_1\}$. During the $i$-th iteration, $c_i$ is chosen such that it maximizes the minimum difference between $c_i$ and observations in $S$:

$$c_i = \arg\max_{x \notin S} \min_{y \in S} C(x, y) \tag{1}$$

where $C(x, y)$ is the DTW difference between $x$ and $y$. The set $S$ is incremented at the end of each iteration such that $S = S \cup c_i$.

The number of clusters discovered by Max-Min clustering is chosen by iterating until the maximized minimum difference falls below a threshold $\theta$. At this point, the set $S$ comprises the centroids for the clustering process, and the final assignment of beats to clusters proceeds by matching each beat to its nearest centroid. Each set of beats assigned to a centroid constitutes a unique cluster. The final number of clusters, $\gamma$, obtained using this process depends on the separability of the underlying data.

The overall effect of the DTW-based partitioning of beats is to transform the original raw ECG signal into a sequence of symbols, i.e., into a sequence of labels corresponding to the different beat morphology classes that occur in the signal. Our approach differs from the methods typically used to annotate ECG signals in two ways. First, we avoid using specialized knowledge to partition beats into known clinical classes. There is a set of generally accepted labels that cardiologists use to differentiate distinct kinds of heart beats. However, in many cases, finer distinctions than provided by these labels can be clinically relevant [7]. Our use of beat clustering rather than beat classification allows us to infer characteristic morphology classes that capture these finer-grained distinctions. Second, our approach does not involve extracting features (e.g., the length of the beat or the amplitude of the P wave) from individual beats. Instead, our clustering algorithm compares the entire raw morphology of pairs of beats. This approach is potentially advantageous, because it does not assume *a priori* knowledge about what aspects of a beat are most relevant. It can also be extended to other time-series data (e.g., blood pressure and respiration waveforms).

## 2.2 Measuring Mismatch in Symbolic Representations

Denoting the set of symbol centroids for patient $p$ as $S_p$ and the set of frequencies with which these symbols occur in the electrocardiogram as $F_p$ (for patient $q$ an analogous representation is adopted), we define the symbolic mismatch between the long-term ECG time-series for patients $p$ and $q$ as:

$$\psi_{p,q} = \sum_{p_i \in S_p} \sum_{q_j \in S_q} C(p_i, q_j) F_p[p_i] F_q[q_j] \tag{2}$$

where $C(p_i, q_j)$ corresponds to the DTW cost of aligning the centroids of symbol classes $p_i$ and $q_j$.

Intuitively, the symbolic mismatch between patients $p$ and $q$ corresponds to an estimate of the expected difference in morphology between any two randomly chosen beats from these patients. The symbolic mismatch computation achieves this by weighting the difference between the centroids for every pair of symbols for the patients by the frequencies with which these symbols occur.

An important feature of symbolic mismatch is that it avoids the need to set up a correspondence between the symbols of patients $p$ and $q$. In contrast to cluster matching techniques [10,11] to compare data for two patients by first making an assignment from symbols in one patient to the other, symbolic mismatch does not require any cross-patient registration of symbols. Instead, it performs weighted morphologic comparisons between all symbol centroids for patients $p$ and $q$. As a result, the symbolization process does not need to be restricted to well-defined labels and is able to use a richer set of patient-specific symbols that capture fine-grained activity over long periods.

## 2.3 Spectrum Clipping and Adaptation for Kernel-based Methods

The formulation for symbolic mismatch in Equation 2 gives rise to a symmetric dissimilarity matrix. For methods that are unable to work directly from dissimilarities, this can be transformed into a similarity matrix using a generalized radial basis function. For both the dissimilarity and similarity case, however, symbolic mismatch can produce a matrix that is indefinite. This can be problematic

when using symbolic mismatch with kernel-based algorithms since the optimization problems become non-convex and the underlying theory is invalidated. In particular, kernel-based classification methods require Mercer's condition to be satisfied by a positive semi-definite kernel matrix [12]. This creates the need to transform the symbolic mismatch matrix before it can be used as a kernel in these methods.

We use spectrum clipping to generalize the use of symbolic mismatch for classification. This approach has been shown both theoretically and empirically to offer advantages over other strategies (e.g., spectrum flipping, spectrum shifting, spectrum squaring, and the use of indefinite kernels) [13]. The symmetric mismatch matrix $\Psi$ has an eigenvalue decomposition:

$$\Psi = U^T \Lambda U \tag{3}$$

where $U$ is an orthogonal matrix and $\Lambda$ is a diagonal matrix of real eigenvalues:

$$\Lambda = diag(\lambda_1, ..., \lambda_n) \tag{4}$$

Spectrum clipping makes $\Psi$ positive semi-definite by clipping all negative eigenvalues to zero. The modified positive semi-definite symbolic mismatch matrix is then given by:

$$\Psi_{clip} = U^T \Lambda_{clip} U \tag{5}$$

where:

$$\Lambda_{clip} = diag(max(\lambda_1, 0), ..., max(\lambda_n, 0)) \tag{6}$$

Using $\Psi_{clip}$ as a kernel matrix is then equivalent to using $(\Lambda_{clip})^{1/2} u_i$ as the $i$-th training sample.

Though we introduce spectrum clipping mainly for the purpose of broadening the applicability of symbolic mismatch to kernel-based methods, this approach offers additional advantages. When the negative eigenvalues of the similarity matrix are caused by noise, one can view spectrum clipping as a denoising step [14]. The results of our experiments, presented later in this paper, support the view of spectrum clipping being useful in a broader context.

## 3  Risk Stratification Using Symbolic Mismatch

We now sketch three different approaches using symbolic mismatch to identify high risk patients in a population. The following two sections contain an empirical evaluation of each. The first approach uses a one-class SVM and a symbolic mismatch similarity matrix obtained using a generalized radial basis transformation. The other two approaches, nearest neighbor analysis and hierarchical clustering, use the symbolic mismatch dissimilarity matrix. In each case, the symbolic mismatch matrix is processed using spectrum clipping.

### 3.1  Classification Approach

SVMs can applied to anomaly detection in a one-class setting [15] . This is done by mapping the data into the feature space corresponding to the kernel and separating instances from the origin with the maximum margin. To separate data from the origin, the following quadratic program is solved:

$$\min_{w, \xi, p} \frac{1}{2} \|w\|^2 + \frac{1}{vn} \sum_i \xi_i - p \tag{7}$$

subject to:

$$(w \cdot \Phi(z_i)) \geq p - \xi_i \ \ i = 1, ..., n \ \ \xi_i \geq 0 \tag{8}$$

where $v$ reflects the tradeoff between incorporating outliers and minimizing the support region.

For a new instance, the label is determined by evaluating which side of the hyperplane the instance falls on in the feature space. The resulting predicted label in terms of the Lagrange multipliers $\alpha_i$ and the spectrum clipped symbolic mismatch similarity matrix $\Psi_{clip}$ is then:

$$\hat{y}_j = sgn(\sum_i \alpha_i \Psi_{clip}(i, j) - p) \tag{9}$$

We apply this approach to train a one-class SVM on all patients. Patients outside the enclosing boundary are labeled anomalies. The parameter $v$ can be varied to control the size of this group.

## 3.2 Nearest Neighbor Approach

Our second approach is based on the concept of nearest neighbor analysis. The assumption underlying this approach is that normal data instances occur in dense neighborhoods, while anomalies occur far from their closest neighbors.

We use an approach similar to [16]. The anomaly score of each patient's long-term time-series is computed as the sum of its distances from the time-series for its $k$-nearest neighbors, as measured by symbolic mismatch. Patients with anomaly scores exceeding a threshold $\theta$ are labeled anomalies.

## 3.3 Clustering Approach

Our third approach is based on hierarchical clustering. We place each patient in a separate cluster, and then proceed in each iteration to merge the two clusters that are most similar to each other. The distance between two clusters is defined as the average of the pairwise symbolic mismatch of the patients in each cluster. The clustering process terminates when it enters the region of diminishing returns (i.e., at the 'knee' of the curve corresponding to the distance of clusters merged together at each iteration). At this point, all patients outside the largest cluster are labeled as anomalies.

# 4 Evaluation Methodology

We evaluated our work on patients enrolled in the DISPERSE2 trial [2]. Patients in the study were admitted to a hospital with non-ST-elevation ACS. Three lead continuous ECG monitoring (LifeCard CF / Pathfinder, DelMar Reynolds / Spacelabs, Issaqua WA) was performed for a median duration of four days at a sampling rate of 128 Hz. The endpoints of cardiovascular death, myocardial infarction and severe recurrent ischemia were adjudicated by a blinded Clinical Events Committee for a median follow-up period of 60 days. The maximum follow-up was 90 days. Data from 686 patients was available after removal of noise-corrupted signals. During the follow-up there were 14 cardiovascular deaths, 28 myocardial infarctions, and 13 cases of severe recurrent ischemia. We define a major adverse cardiac event to be any of these three adverse events.

We studied the effectiveness of combining symbolic mismatch with each of classification, nearest neighbor analysis and clustering in identifying a high risk group of patients. Consistent with other clinical studies to evaluate methods for risk stratification in the setting of ACS [17], we classified patients in the highest quartile as the high risk group. For the classification approach, this corresponded to choosing $v$ such that the group of patients lying outside the enclosing boundary constituted roughly 25% of the population. For the nearest neighbor approach we investigated all odd values of $k$ from 3 to 9, and patients with anomaly scores in the top 25% of the population were classified as being at high risk. For the clustering approach, the varying sizes of the clusters merged together at each step made it difficult to select a high risk quartile. Instead, patients lying outside the largest cluster were categorized as being at risk. In the tests reported later in this paper, this group contained roughly 23% the patients in the population. We used the LIBSVM implementation for our one-class SVM. Both the nearest neighbor and clustering approaches were carried out using MATLAB implementations.

We employed Kaplan-Meier survival analysis to compare the rates for major adverse cardiac events between patients declared to be at high and low risk. Hazard ratios (HR) and 95% confidence interval (CI) were estimated using a Cox proportional hazards regression model. The predictions of each approach were studied in univariate models, and also in multivariate models that additionally included other clinical risk variables (age$\geq$65 years, gender, smoking history, hypertension, diabetes mellitus, hyperlipidemia, history of chronic obstructive pulmonary disorder (COPD), history of coronary heart disease (CHD), previous MI, previous angina, ST depression on admission, index diagnosis of MI) as well as ECG risk metrics proposed in the past (heart rate variability (HRV), heart rate turbulence (HRT), and deceleration capacity (DC)) [18].

| Method | HR | P Value | 95% CI |
|---|---|---|---|
| One-Class SVM | 1.38 | 0.033 | 1.04-1.89 |
| 3-Nearest Neighbor | 1.91 | 0.031 | 1.06-3.44 |
| 5-Nearest Neighbor | 2.10 | 0.013 | 1.17-3.76 |
| 7-Nearest Neighbor | 2.28 | 0.005 | 1.28-4.07 |
| 9-Nearest Neighbor | 2.07 | 0.015 | 1.15-3.71 |
| Hierarchical Clustering | 2.04 | 0.017 | 1.13-3.68 |

Table 1: Univariate association of risk predictions from different approaches using symbolic mismatch with major adverse cardiac events over a 90 day period following ACS.

| Clinical Variable | HR | P Value | 95% CI |
|---|---|---|---|
| Age≥65 years | 1.82 | 0.041 | 1.02-3.24 |
| Female Gender | 0.69 | 0.261 | 0.37-1.31 |
| Current Smoker | 1.05 | 0.866 | 0.59-1.87 |
| Hypertension | 1.44 | 0.257 | 0.77-2.68 |
| Diabetes Mellitus | 1.95 | 0.072 | 0.94-4.04 |
| Hyperlipidemia | 1.00 | 0.994 | 0.55-1.82 |
| History of COPD | 1.05 | 0.933 | 0.37-2.92 |
| History of CHD | 1.10 | 0.994 | 0.37-2.92 |
| Previous MI | 1.17 | 0.630 | 0.62-2.22 |
| Previous angina | 0.94 | 0.842 | 0.53-1.68 |
| ST depression>0.5mm | 1.13 | 0.675 | 0.64-2.01 |
| Index diagnosis of MI | 1.42 | 0.134 | 0.90-2.26 |
| Heart Rate Variability | 1.56 | 0.128 | 0.88-2.77 |
| Heart Rate Turbulence | 1.64 | 0.013 | 1.11-2.42 |
| Deceleration Capacity | 1.77 | 0.002 | 1.23-2.54 |

Table 2: Univariate association of existing clinical and ECG risk variables with major adverse cardiac events over a 90 day period following ACS.

## 5 Results

### 5.1 Univariate Results

Results of univariate analysis for all three unsupervised symbolic mismatch-based approaches are presented in Table 1. The predictions from all methods showed a statistically significant (i.e., $p < 0.05$) association with major adverse cardiac events following ACS. The results in Table 1 can be interpreted as roughly a doubled rate of adverse outcomes per unit time in patients identified as being at high risk by the nearest neighbor and clustering approaches. For the classification approach, patients identified as being at high risk had a nearly 40% increased risk.

For comparison, we also include the univariate association of the other clinical and ECG risk variables in our study (Table 2). Both the nearest neighbor and clustering approaches had a higher hazard ratio in this patient population than any of the other variables studied. Of the clinical risk variables, only age was found to be significantly associated on univariate analysis with major cardiac events after ACS. Diabetes ($p$=0.072) was marginally outside the 5% level of significance. Of the ECG risk variables, both HRT and DC showed a univariate association with major adverse cardiac events in this population. These results are consistent with the clinical literature on these risk metrics.

### 5.2 Multivariate Results

We measured the correlation between the predictions of the unsupervised symbolic mismatch-based approaches and both the clinical and ECG risk variables. All of the unsupervised approaches had low correlation with both sets of variables ($R \leq 0.2$). This suggests that the results of these novel approaches can be usefully combined with results of existing approaches.

On multivariate analysis, both the nearest neighbor approach and the clustering approach were independent predictors of adverse outcomes (Table 3). In our study, the nearest neighbor approach (for $k > 3$) had the highest hazard ratio on both univariate and multivariate analysis. Both the nearest neighbor and clustering approaches predicted patients with an approximately two-fold increased risk of adverse outcomes. This increased risk did not change much even after adjusting for other clinical and ECG risk variables.

| Method | Adjusted HR | P Value | 95% CI |
|---|---|---|---|
| One-Class SVM | 1.32 | 0.074 | 0.97-1.79 |
| 3-Nearest Neighbor | 1.88 | 0.042 | 1.02-3.46 |
| 5-Nearest Neighbor | 2.07 | 0.018 | 1.13-3.79 |
| 7-Nearest Neighbor | 2.25 | 0.008 | 1.23-4.11 |
| 9-Nearest Neighbor | 2.04 | 0.021 | 1.11-3.73 |
| Hierarchical Clustering | 1.86 | 0.042 | 1.02-3.46 |

Table 3: Multivariate association of high risk predictions from different approaches using symbolic mismatch with major adverse cardiac events over a 90 day period following ACS. Multivariate results adjusted for variables in Table 2.

| Method | HR | P Value | 95% CI |
|---|---|---|---|
| One-Class SVM | 1.36 | 0.038 | 1.01-1.79 |
| 3-Nearest Neighbor | 1.74 | 0.069 | 0.96-3.16 |
| 5-Nearest Neighbor | 1.57 | 0.142 | 0.86-2.88 |
| 7-Nearest Neighbor | 1.73 | 0.071 | 0.95-3.14 |
| 9-Nearest Neighbor | 1.89 | 0.034 | 1.05-3.41 |
| Hierarchical Clustering | 1.19 | 0.563 | 0.67-2.12 |

Table 4: Univariate association of high risk predictions without the use of spectrum clipping. None of the approaches showed a statistically significant association with the study endpoint in any of the multivariate models including other clinical risk variables when spectrum clipping was not used.

## 5.3   Effect of Spectrum Clipping

We also investigated the effect of spectrum clipping on the performance of our different risk stratification approaches. Table 4 presents the associations when spectrum clipping was not used. For all three methods, performance was worse without the use of spectrum clipping, although the effect was small for the one-class SVM case.

## 6   Related Work

Most previous work on comparing signals in terms of their raw samples (e.g., metrics such as dynamic time warping, longest common subsequence, edit distance with real penalty, sequence weighted alignment, spatial assembling distance, threshold queries) [19] focuses on relatively short time-series. This is due to the runtime of these methods (quadratic for many methods) and the need to reason in terms of the frequency and dynamics of higher-level signal constructs (as opposed to individual samples) when studying systems over long periods.

Most prior research on comparing long-term time-series focuses instead on extracting specific features from long-term signals and quantifying the differences between these features. In the context of cardiovascular disease, long-term ECG is often reduced to features (e.g., mean heart rate or heart rate variability) and compared in terms of these features. These approaches, unlike our symbolic mismatch based approaches, draw upon significant *a priori* knowledge. Our belief was that for applications like risk stratifying patients for major cardiac events, focusing on a set of specialized features leads to important information being potentially missed. In our work, we focus instead on developing an approach that avoids use of significant *a priori* knowledge by comparing the raw morphology of long-term time-series. We propose doing this in a computationally efficient and systematic way through symbolization. While this use of symbolization represents a lossy compression of the original signal, the underlying DTW-based process of quantifying differences between long-term time-series remains grounded in the comparison of raw morphology.

Symbolization maps the comparison of long-term time-series into the domain of sequence comparison. There is an extensive body of prior work focusing on the comparison of sequential or string data. Algorithms based on measuring the edit distance between strings are widely used in disciplines such as computational biology, but are typically restricted to comparisons of short sequences because of their computational complexity. Research on the use of profile hidden Markov models [20,21] to optimize recognition of binary labeled sequences is more closely related to our work. This work focuses on learning the parameters of a hidden Markov model that can represent approximations of sequences and can be used to score other sequences. Such approaches require large amounts of data or good priors to train the hidden Markov models. Computing forward and backward prob-

abilities from the Baum-Welch algorithm is also very computationally intensive. Other research in this area focuses on mismatch tree-based kernels [22], which use the mismatch tree data structure [23] to quantify the difference between two sequences based on the approximate occurrence of fixed length subsequences within them. Similar to this approach is work on using a "bag of motifs" representation [24], which provides a more flexible representation than fixed length subsequences but usually requires prior knowledge of motifs in the data [24].

In contrast to these efforts, we use a simple computationally efficient approach to compare symbolic sequences without prior knowledge. Most importantly, our approach helps address the situation where symbolizing long-term time-series in a patient-specific manner leads to the symbolic sequences from different alphabets [25]. In this case, hidden Markov models, mismatch trees or a "bag of motifs" approach trained on one patient cannot be easily used to score the sequences for other patients. Instead, any comparative approach must maintain a hard or soft registration of symbols across individuals. Symbolic mismatch complements existing work on sequence comparison by using a measure that quantifies differences across patients while retaining information on how the symbols for these patients differ.

Finally, we distinguish our work from earlier method for ECG-based risk stratification. These methods typically calculate a particular pre-defined feature from the raw ECG signal, and to use it to rank patients along a risk continuum. Our approach, focusing on detecting patients with high symbolic mismatch relative to other patients in the population, is orthogonal to the use of specialized high risk features along two important dimensions. First, it does not require the presence of significant prior knowledge. For the cardiovascular care, we only assume that ECG signals from patients who are at high risk differ from those of the rest of the population. There are no specific assumptions about the nature of these differences. Second, the ability to partition patients into groups with similar ECG characteristics and potentially common risk profiles potentially allows for a more fine-grained understanding of a how a patient's future health may evolve over time. Matching patients to past cases with similar ECG signals could lead to more accurate assignments of risk scores for particular events such as death and recurring heart attacks.

## 7 Discussion

In this paper, we described a novel unsupervised learning approach to cardiovascular risk stratification that is complementary to existing clinical approaches.

We proposed using symbolic mismatch to quantify differences in long-term physiological time-series. Our approach uses a symbolic transformation to measure changes in the morphology and frequency of prototypical functional units observed over long periods in two signals. Symbolic mismatch avoids feature extraction and deals with inter-patient differences in a parameter-less way. We also explored the hypothesis that high risk patients in a population can be identified as individuals with anomalous long-term signals. We developed multiple comparative approaches to detect such patients, and evaluated these methods in a real-world application of risk stratification for major adverse cardiac events following ACS.

Our results suggest that symbolic mismatch-based comparative approaches may have clinical utility in identifying high risk patients, and can provide information that is complementary to existing clinical risk variables. In particular, we note that the hazard ratios we report are typically considered clinically meaningful. In a different study of 118 variables in 15,000 post-ACS patients with 90 day follow-up similar to our population, [1] did not find any variables with a hazard ratio greater than 2.00. We observed a similar result in our patient population, where all of the existing clinical and ECG risk variables had a hazard ratio less than 2.00. In contrast to this, our nearest neighbor-based approach achieved a hazard ratio of 2.28, even after being adjusted for existing risk measures.

Our study has limitations. While our decision to compare the morphology and frequency of prototypical functional units leads to a measure that is computationally efficient on large volumes of data, this process does not capture information related to the dynamics of these prototypical units. We also observe that all three of the comparative approaches investigated in our study focus only on identifying patients who are anomalies. While we believe that symbolic mismatch may have further use in supervised learning, this hypothesis needs to be evaluated more fully in future work.

# References

[1] LK Newby, MV Bhapkar, HD White et al. (2003) Predictors of 90-day outcome in patients stabilized after acute coronary syndromes. *Eur Heart J*, 172-181.

[2] C.P. Cannon, S. Husted, R.A. Harringtonet al. (2007) Safety, Tolerability, and Initial Efficacy of AZD6140, the First Reversible Oral Adenosine Diphosphate Receptor Antagonist, Compared With Clopidogrel, in Patients With NonST-Segment Elevation Acute Coronary Syndrome Primary. *J Am Coll Cardiol*, 1844-1851.

[3] World Health Organization. (2009) Cardiovascular Diseases Fact Sheet.

[4] J. Mackay, G.A. Mensah, S. Mendis et al. (2004) *The Atlas of Heart Disease and Stroke*. WHO.

[5] J.J. Bailey, A.S. Berson, H. Handelsman et al. (2001) Utility of current risk stratification tests for predicting major arrhythmic events after myocardial infarction. *J Am Coll Cardio*, 1902-1911.

[6] G. Lopera & A.B. Curtis. (2009) Risk stratification for sudden cardiac death: current approaches and predictive value. *Curr Cardiol Rev*, 56-64.

[7] Z. Syed, J. Guttag & C. Stultz. (2007) Clustering and Symbolic Analysis of Cardiovascular Signals: Discovery and Visualization of Medically Relevant Patterns in Long-Term Data Using Limited Prior Knowledge. *EURASIP J Adv Sig Proc*, 1-16.

[8] P.S. Hamilton & W.J. Tompkins. (1986) Quantitative investigation of QRS detection rules using the MIT/BIH arrhythmia database. *IEEE Trans Biomed Eng*, 1157-1165.

[9] W. Zong, GB Moody, & D. Jiang. (2003) A robust open-source algorithm to detect onset and duration of QRS complexes. *Comp Cardiol*, 737-740.

[10] S.H. Chang, F.H. Cheng, W. Hsu et al. (1997) Fast algorithm for point pattern matching: invariant to translations, rotations and scale changes. *Pattern Recognition*, 311-320.

[11] W.W. Cohen & J. Richman (2002). Learning to match and cluster large high-dimensional data sets for data integration. In *Proc. ACM SIGKDD*, 475-480.

[12] B. Scholkopf & A.J. Smola. (2002) *Learning with Kernels*. MIT Press.

[13] Y. Chen, E.K. Garcia, M.R. Gupta et al. (2009) Similarity-based classification: concepts and algorithms. *JMLR*, 747-776.

[14] G. Wu, EY. Chang & Z. Zhang. (2005) An analysis of transformation on non-positive semidefinite similarity matrix for kernel machines. Technical report, University of California, Santa Barbara.

[15] B. Scholkopf, J.C. Platt, J. Shawe-Taylor, et al. (2001) Estimating the support of a high-dimensional distribution. *Neural Computation*, 1443-1471.

[16] E. Eskin, A. Arnold, M. Prerau et al. (2002) A geometric framework for unsupervised anomaly detection. *App Data Mining Comp Secur*, 1-20.

[17] M.G. Shlipak, J.H. Ix, K. Bibbins-Domingo et al. (2008) Biomarkers to predict recurrent cardiovascular disease: the Heart and Soul Study. *JAMA*, 50-57.

[18] B. M. Scirica. (2010) Acute coronary syndrome: emerging tools for diagnosis and risk assessment. *J Am Coll Cardiol*, 1403-1415.

[19] H. Ding, G. Trajcevski, P Scheuermann et al. (2008) Querying and mining of time series data: experimental comparison of representations and distance measures. In *Proc. VLDB*, 1542-1552.

[20] A. Krogh. (1994) Hidden Markov models for labeled sequences. In *Proc. ICPR*, 140-144.

[21] T. Jaakkola, M. Diekhans & D. Haussler. (1999) Using the Fisher kernel method to detect remote protein homologies. In *Proc. ICISMB*, 149-158.

[22] C. Leslie, E. Eskin, J. Weston et al. (2003) Mismatch string kernels for SVM protein classification. In *Proc. NIPS*, 1441-1448.

[23] E. Eskin & P.A. Pevzner. (2002) Finding composite regulatory patterns in DNA sequences. *Bioinformatics*, 354-363.

[24] A. Ben-Hur & D. Brutlag. (2006) Sequence motifs: highly predictive features of protein function. *Feature Extraction*, 625-645.

[25] Z. Syed, C. Stultz, M. Kellis et al. (2010) Motif discovery in physiological datasets: a methodology for inferring predictive elements. *ACM Trans. Knowledge Discovery in Data*, 1-23.

